# Generalization in Clustering with Unobserved Features

**Eyal Krupka   and   Naftali Tishby**
School of Computer Science and Engineering,
Interdisciplinary Center for Neural Computation
The Hebrew University Jerusalem, 91904, Israel
{eyalkr,tishby}@cs.huji.ac.il

## Abstract

We argue that when objects are characterized by many attributes, clustering them on the basis of a relatively small *random* subset of these attributes can capture information on the unobserved attributes as well. Moreover, we show that under mild technical conditions, clustering the objects on the basis of such a random subset performs almost as well as clustering with the full attribute set. We prove a finite sample generalization theorems for this novel learning scheme that extends analogous results from the supervised learning setting. The scheme is demonstrated for collaborative filtering of users with movies rating as attributes.

## 1   Introduction

Data clustering is unsupervised classification of objects into groups based on their similarity [1]. Often, it is desirable to have the clusters to match some labels that are unknown to the clustering algorithm. In this context, a good data clustering is expected to have homogeneous labels in each cluster, under some constraints on the number or complexity of the clusters. This can be quantified by mutual information (see e.g. [2]) between the objects' cluster identity and their (unknown) labels, for a given complexity of clusters. Since the clustering algorithm has no access to the labels, it is unclear how the algorithm can optimize the quality of the clustering. Even worse, the clustering quality depends on the specific choice of the unobserved labels. For example a good documents clustering with respect to topics is very different from a clustering with respect to authors.

In our setting, instead of trying to cluster by some "arbitrary" labels, we try to predict unobserved features from observed ones. In this sense our target "labels" are yet other features that "happened" to be unobserved. For example, when clustering fruits based on their observed features, such as shape, color and size, the target of clustering is to match unobserved features, such as nutritional value and toxicity.

In order to theoretically analyze and quantify this new learning scheme, we make the following assumptions. Consider an infinite set of features, and assume that we observe only a *random* subset of $n$ features, called *observed features*. The other features are called *unobserved features*. We assume that the random selection of features is done uniformly and independently.

Table 1: Analogy with supervised learning

| Training set | $n$ randomly selected features (observed features) |
|---|---|
| Test set | Unobserved features |
| Learning algorithm | Cluster the *instances* into $k$ clusters |
| Hypothesis class | All possible partitions of $m$ instances into $k$ clusters |
| Min generalization error | Max expected information on *unobserved* features |
| ERM | Observed Information Maximization (OIM) |
| Good generalization | Mean *observed* and *unobserved* information are similar |

The clustering algorithm has access only to the observed features of $m$ instances. After the clustering, one of the *unobserved* features is randomly and uniformly selected to be a target label, i.e. clustering performance is measured with respect to this feature. Obviously, the clustering algorithm cannot be directly optimized for this specific feature.

The question is whether we can optimize the *expected* performance on the unobserved feature, based on the observed features alone. The expectation is over the *random* selection of the target feature. In other words, can we find clusters that match as many unobserved features as possible? Perhaps surprisingly, for large enough number of observed features, the answer is yes. We show that for any clustering algorithm, the average performance of the clustering with respect to the observed and unobserved features, is similar. Hence we can indirectly optimize clustering performance with respect to the unobserved features, in analogy to generalization in supervised learning. These results are universal and do not require any additional assumptions such as underling model or a distribution that created the instances.

In order to quantify these results, we define two terms: the average observed information and the expected unobserved information. Let $T$ be the variable which represents the cluster for each instance, and $\{X_1, ..., X_\infty\}$ the set of random variables which denotes the features. The average observed information, denoted by $I_{ob}$, is the average mutual information between $T$ and each of the observed features. In other words, if the observed features are $\{X_1, ..., X_n\}$ then $I_{ob} = \frac{1}{n} \sum_{j=1}^{n} I(T; X_j)$. The expected unobserved information, denoted by $I_{un}$, is the *expected* value of the mutual information between $T$ and a *randomly* selected unobserved feature, i.e. $E_j\{I(T; X_j)\}$. Note that whereas $I_{ob}$ can be measured directly, this paper deals with the question of how to infer and maximize $I_{un}$.

Our main results consist of two theorems. The first is a generalization theorem. It gives an upper bound on the probability of large difference between $I_{ob}$ and $I_{un}$ for all possible clusterings. It also states a *uniform convergence in probability* of $|I_{ob} - I_{un}|$ as the number of observed features increases. Conceptually, the observed mean information, $I_{ob}$, is analogous to the training error in standard supervised learning [3], whereas the unobserved information, $I_{un}$, is similar to the generalization error.

The second theorem states that under constraint on the number of clusters, and large enough number of observed features, one can achieve nearly the best possible performance, in terms of $I_{un}$. Analogous to the principle of Empirical Risk Minimization (ERM) in statistical learning theory [3], this is done by maximizing $I_{ob}$.

Table 1 summarizes the correspondence of our setting to that of supervised learning. The key difference is that in supervised learning, the set of features is fixed and the training instances (samples) are assumed to be randomly drawn from some distribution. In our setting, the set of instances is fixed, but the set of observed features is assumed to be randomly selected.

Our new theorems are evaluated empirically in section 3, on a data set of movie ratings.

This empirical test also suggests one future research direction: use the framework suggested in this paper for collaborative filtering. Our main point in this paper, however, is the new conceptual framework and not a specific algorithm or experimental performance.

**Related work** The idea of an information tradeoff between complexity and information on target variables is similar to the idea of the information bottleneck [4]. But unlike the bottleneck method, here we are trying to maximize information on *unobserved* variables, using finite samples.

In the framework of learning with labeled and unlabeled data [5], a fundamental issue is the link between the marginal distribution $P(\mathbf{x})$ over examples $\mathbf{x}$ and the conditional $P(y|\mathbf{x})$ for the label $y$ [6]. From this point of view our approach assumes that $y$ is a feature in itself.

## 2 Mathematical Formulation and Analysis

Consider a set of discrete random variables $\{X_1, ..., X_L\}$, where L is very large ($L \to \infty$). We randomly, uniformly and independently select $n << \sqrt{L}$ variables from this set. These variables are the observed features and their indexes are denoted by $\{q_1, ..., q_n\}$. The remaining $L - n$ variables are the *unobserved features*. A clustering algorithm has access only to the *observed* features over $m$ instances $\{\mathbf{x}[1], ..., \mathbf{x}[m]\}$. The algorithm assigns a cluster label $t_i \in \{1, ..., k\}$ for each instance $\mathbf{x}[i]$, where $k$ is the number of clusters. Let $T$ denote the cluster label assigned by the algorithm.

Shannon's mutual information between two variables is a function of their joint distribution, defined as $I(T; X_j) = \sum_{t,x_j} P(t, x_j) \log \left( \frac{P(t,x_j)}{P(t)P(x_j)} \right)$. Since we are dealing with a finite number of samples, $m$, the distribution $P$ is taken as the *empirical* joint distribution of $(T, X_j)$, for every $j$. For a random $j$, this empirical mutual information is a random variable on its own.

The average observed information, $I_{ob}$, is now defined as $I_{ob} = \frac{1}{n} \sum_{i=1}^n I(T; X_{q_i})$. In general, $I_{ob}$ is higher when clusters are more coherent, i.e. elements within each cluster have many similar attributes. The expected unobserved information, $I_{un}$, is defined as $I_{un} = E_j \{I(T; X_j)\}$. We can assume that the unobserved feature is with high probability from the unobserved set. Equivalently, $I_{un}$ can be the mean mutual information between the clusters and each of the unobserved features, $I_{un} = \frac{1}{L-n} \sum_{j \notin \{q_1,...,q_n\}} I(T; X_j)$.

The goal of the clustering algorithm is to find cluster labels $\{t_1, ..., t_m\}$, that maximize $I_{un}$, subject to a constraint on their complexity - henceforth considered as the number of clusters ($k \leq D$) for simplicity, where $D$ is an integer bound.

Before discussing how to maximize $I_{un}$, we consider first the problem of estimating it. Similar to the generalization error in supervised learning, $I_{un}$ cannot be estimated directly in the learning algorithm, but we may be able to bound the difference between the observed information $I_{ob}$ - our "training error" - and $I_{un}$ - the "generalization error". To obtain generalization this bound should be *uniform over all possible clusterings* with a high probability over the randomly selected features. The following lemma argues that such *uniform convergence in probability* of $I_{ob}$ to $I_{un}$ always occurs.

**Lemma 1** *With the definitions above,*

$$\Pr \left\{ \sup_{\{t_1,...,t_m\}} |I_{ob} - I_{un}| > \epsilon \right\} \leq 2e^{-2n\epsilon^2/(\log k)^2 + m \log k} \quad \forall \epsilon > 0$$

*where the probability is over the random selection of the observed features.*

**Proof:** For fixed cluster labels, $\{t_1, ..., t_m\}$, and a random feature $j$, the mutual information $I(T; X_j)$ is a function of the random variable $j$, and hence $I(T; X_j)$ is a random variable in itself. $I_{ob}$ is the average of $n$ such independent random variables and $I_{un}$ is its expected value. Clearly, for all $j$, $0 \leq I(T; X_j) \leq \log k$. Using Hoeffding's inequality [7], $\Pr\left\{|I_{ob} - I_{un}| > \epsilon\right\} \leq 2e^{-2n\epsilon^2/(\log k)^2}$. Since there are at most $k^m$ possible partitions, the union bound is sufficient to prove the lemma 1. □

Note that for any $\epsilon > 0$, the probability that $|I_{ob} - I_{un}| > \epsilon$ goes to zero, as $n \to \infty$. The convergence rate of $I_{ob}$ to $I_{un}$ is bounded by $O(\log n / \sqrt{n})$. As expected, this upper bound decreases as the number of clusters, $k$, decreases.

Unlike the standard bounds in supervised learning, this bound increases with the number of instances ($m$), and decreases with increasing number of observed features ($n$). This is because in our scheme the training size is not the number of instances, but rather the number of observed features (See Table 1). However, in the next theorem we obtain an upper bound that is independent of $m$, and hence is tighter for large $m$.

**Theorem 1** *(Generalization Theorem) With the definitions above,*

$$\Pr\left\{\sup_{\{t_1,...,t_m\}} |I_{ob} - I_{un}| > \epsilon\right\} \leq 8(\log k)e^{-\frac{n\epsilon^2}{8(\log k)^2} + \frac{4k \max_j |\mathcal{X}_j|}{\epsilon}\log k - \log \epsilon} \quad \forall \epsilon > 0$$

*where $|\mathcal{X}_j|$ denotes the alphabet size of $X_j$ (i.e. the number of different values it can obtain). Again, the probability is over the random selection of the observed features.*

The convergence rate here is bounded by $O(\log n / ^3\sqrt{n})$. However, for relatively large $n$ one can use the bound in lemma 1, which converge faster.

A detailed proof of theorem 1 can be found in [8]. Here we provide the outline of the proof.

**Proof outline:** From the given $m$ instances and any given cluster labels $\{t_1, ..., t_m\}$, draw uniformly and independently $m'$ instances (repeats allowed) and denote their indexes by $\{i_1, ..., i_{m'}\}$. We can estimate $I(T; X_j)$ from the empirical distribution of $(T, X_j)$ over the $m'$ instances. This distribution is denoted by $\hat{P}(t, x_j)$ and the corresponding mutual information is denoted by $I_{\hat{P}}(T; X_j)$. Theorem 1 is build up from the following upper bounds, which are independent of $m$, but depend on the choice of $m'$. The first bound is on $E\left\{\left|I(T; X_j) - I_{\hat{P}}(T; X_j)\right|\right\}$, where the expectation is over random selection of the $m'$ *instances*. From this bound we derive upper bounds on $|I_{ob} - E(\hat{I}_{ob})|$ and $|I_{un} - E(\hat{I}_{un})|$, where $\hat{I}_{ob}$, $\hat{I}_{un}$ are the estimated values of $I_{ob}$, $I_{un}$ based on the subset of $m'$ instances. The last required bound is on the probability that $\sup_{\{t_1,...,t_m\}} |E(\hat{I}_{ob}) - E(\hat{I}_{un})| > \epsilon_1$, for any $\epsilon_1 > 0$. This bound is obtained from lemma 1. The choice of $m'$ is independent on $m$. Its value should be large enough for the estimations $\hat{I}_{ob}$, $\hat{I}_{un}$ to be accurate, but not too large, so as to limit the number of possible clusterings over the $m'$ instances.

We now describe the above mentioned upper bounds in more details. Using Paninski [9] (proposition 1) it is easy to show that the bias between $I(T; X_j)$ and its maximum likelihood estimation, based on $\hat{P}(t, x_j)$ is bounded as follows.

$$E_{\{i_1,...,i_{m'}\}}\left\{\left|I(T; X_j) - I_{\hat{P}}(T; X_j)\right|\right\} \leq \log\left(1 + \frac{k|\mathcal{X}_j| - 1}{m'}\right) \leq \frac{k|\mathcal{X}_j|}{m'} \qquad (1)$$

From this equation we obtain,

$$|I_{ob} - E_{\{i_1,...,i_{m'}\}}(\hat{I}_{ob})|, \; |I_{un} - E_{\{i_1,...,i_{m'}\}}(\hat{I}_{un})| \leq k \max_j |\mathcal{X}_j| / m' \qquad (2)$$

Using lemma 1 we have an upper bound on the probability that $\sup_{\{t_1,...,t_m\}} |\hat{I}_{ob} - \hat{I}_{un}| > \epsilon$ over the random selection of *features*, as a function of $m'$. However, the upper bound we need is on the probability that $\sup_{\{t_1,...,t_m\}} |E(\hat{I}_{ob}) - E(\hat{I}_{un})| > \epsilon_1$. Note that the expectations $E(\hat{I}_{ob})$, $E(\hat{I}_{un})$ are done over random selection of the subset of $m'$ *instances*, for a set of features that were randomly selected *once*. In order to link between these two probabilities, we need the following lemma.

**Lemma 2** *Consider a function $f$ of two independent random variables $(Y, Z)$. We assume that $f(y, z) \leq c$, $\forall y, z$, where $c$ is some constant. If $\Pr\{f(Y, Z) > \tilde{\epsilon}\} \leq \delta$, then*

$$\Pr_Z \{E_y (f(y, Z)) \geq \epsilon\} \leq \frac{c - \tilde{\epsilon}}{\epsilon - \tilde{\epsilon}}\delta \quad \forall \epsilon > \tilde{\epsilon}$$

The proof of this lemma is rather standard and is given in [8]. From lemmas 1 and 2 it is easy to show that

$$\Pr\left\{E_{\{i_1,...,i_{m'}\}}\left(\sup_{\{t_1,...,t_m\}} \left|\hat{I}_{ob} - \hat{I}_{un}\right|\right) > \epsilon_1\right\} \leq \frac{4\log k}{\epsilon_1}e^{-\frac{n\epsilon_1^2}{2(\log k)^2} + m'\log k} \quad (3)$$

Lemma 2 is used, where $Z$ represents the random selection of features, $Y$ represents the random selection of $m'$ instances, $f(y, z) = \sup_{\{t_1,...,t_m\}} |\hat{I}_{ob} - \hat{I}_{un}|$, $c = \log k$, and $\tilde{\epsilon} = \epsilon_1/2$. From eq. 2 and 3 it can be shown that

$$\Pr\left\{\sup_{\{t_1,...,t_m\}} |I_{ob} - I_{un}| > \epsilon_1 + \frac{2k\max_j |\mathcal{X}_j|}{m'}\right\} \leq \frac{4\log k}{\epsilon_1}e^{-\frac{n\epsilon_1^2}{2(\log k)^2} + m'\log k}$$

By selecting $\epsilon_1 = \epsilon/2$, $m' = 4k\max_j |\mathcal{X}_j|/\epsilon$, we obtain theorem 1. □

Note that the selection of $m'$ depends on $k\max_j |\mathcal{X}_j|$. This reflects the fact that in order to accurately estimate $I(T, X_j)$, we need a number of instances, $m'$, which is much larger than the product of the alphabet sizes of $T$, $X_j$.

We can now return to the problem of specifying a clustering that maximizes $I_{un}$, using only the observed features. For a reference, we will first define $I_{un}$ of the best possible clusters.

**Definition 1** *Maximally achievable unobserved information: Let $I_{un,D}^*$ be the maximum value of $I_{un}$ that can be achieved by any clustering $\{t_1, ..., t_m\}$, subject to the constraint $k \leq D$, for some constant $D$*

$$I_{un,D}^* = \sup_{\{\{t_1,...,t_m\}:k\leq D\}} I_{un}$$

*The clustering that achieves this value is called **the best clustering**. The average observed information of this clustering is denoted by $I_{ob,D}^*$.*

**Definition 2** *Observed information maximization algorithm: Let **IobMax** be any clustering algorithm that, based on the values of observed features alone, selects the cluster labels $\{t_1, ..., t_m\}$ having the maximum possible value of $I_{ob}$, subject to the constraint $k \leq D$.*

*Let $\tilde{I}_{ob,D}$ be the average observed information achieved by IobMax algorithm. Let $\tilde{I}_{un,D}$ be the expected unobserved information achieved by the IobMax algorithm.*

The next theorem states that *IobMax* not only maximizes $I_{ob}$, but also $I_{un}$.

**Theorem 2** *With the definitions above,*

$$\Pr\left\{\tilde{I}_{un,D} \leq I^*_{un,D} - \epsilon\right\} \leq 8(\log k)e^{-\frac{n\epsilon^2}{32(\log k)^2} + \frac{8k\max_j |\mathcal{X}_j|}{\epsilon}\log k - \log(\epsilon/2)} \quad \forall \epsilon > 0 \quad (4)$$

*where the probability is over the random selection of the observed features.*

**Proof:** We now define a *bad clustering* as a clustering whose expected unobserved information satisfies $I_{un} \leq I^*_{un,D} - \epsilon$. Using Theorem 1, the probability that $|I_{ob} - I_{un}| > \epsilon/2$ for any of the clusterings is upper bounded by the right term of equation 4. If for all clusterings $|I_{ob} - I_{un}| \leq \epsilon/2$, then surely $I^*_{ob,D} \geq I^*_{un,D} - \epsilon/2$ (see Definition 1) and $I_{ob}$ of all bad clusterings satisfies $I_{ob} \leq I^*_{un,D} - \epsilon/2$. Hence the probability that a bad clustering has a higher average observed information than the best clustering is upper bounded as in Theorem 2. □

As a result of this theorem, when $n$ is large enough, even an algorithm that knows the value of *all* the features (observed and unobserved) cannot find a clustering with the same complexity ($k$) which is significantly better than the clustering found by $IobMax$ algorithm.

## 3 Empirical Evaluation

In this section we describe an experimental evaluation of the generalization properties of the *IobMax* algorithm for a finite large number of features. We examine the difference between $I_{ob}$ and $I_{un}$ as function of the number of observed features and the number of clusters used. We also compare the value of $I_{un}$ achieved by *IobMax* algorithm to the maximum achievable $I^*_{un,D}$ (See definition 1).

Our evaluation uses a data set typically used for collaborative filtering. Collaborative filtering refers to methods of making predictions about a user's preferences, by collecting preferences of many users. For example, collaborative filtering for movie ratings could make predictions about rating of movies by a user, given a partial list of ratings from this user and many other users. Clustering methods are used for collaborative filtering by cluster users based on the similarity of their ratings (see e.g. [10]).

In our setting, each user is described as a vector of movie ratings. The rating of each movie is regarded as a feature. We cluster users based on the set of observed features, i.e. rated movies. In our context, the goal of the clustering is to maximize the information between the clusters and unobserved features, i.e. movies that have not yet been rated by any of the users. By Theorem 2, given large enough number of rated movies, we can achieve the best possible clustering of users with respect to unseen movies. In this region, no additional information (such as user age, taste, rating of more movies) beyond the observed features can improve $I_{un}$ by more than some small $\epsilon$.

The purpose of this section is *not* to suggest a new algorithm for collaborative filtering or compare it to other methods, but simply to illustrate our new theorems on empirical data.

**Dataset.** We used MovieLens (www.movielens.umn.edu), which is a movie rating data set. It was collected distributed by GroupLens Research at the University of Minnesota. It contains approximately 1 million ratings for 3900 movies by 6040 users. Ratings are on a scale of 1 to 5. We used only a subset consisting of 2400 movies by 4000 users. In our setting, each instance is a vector of ratings $(x_1, ..., x_{2400})$ by specific user. Each movie is viewed as a feature, where the rating is the value of the feature.

**Experimental Setup.** We randomly split the 2400 movies into two groups, denoted by "A" and "B", of 1200 movies (features) each. We used a subset of the movies from group "A" as observed features and all movies from group "B" as the unobserved features. The experiment was repeated with 10 random splits and the results averaged. We estimated $I_{un}$ by the mean information between the clusters and ratings of movies from group "B".

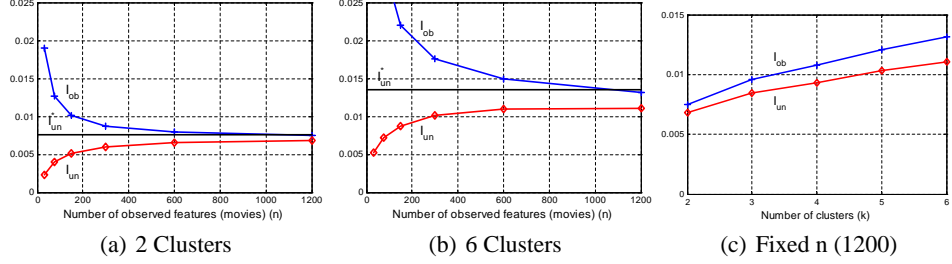

| (a) 2 Clusters | (b) 6 Clusters | (c) Fixed n (1200) |

Figure 1: $I_{ob}$, $I_{un}$ and $I_{un}^*$ per number of training movies and clusters. In (a) and (b) the number of movies is variable, and the number of clusters is fixed. In (c) The number of observed movies is fixed (1200), and the number of clusters is variable. The overall mean information is low, since the rating matrix is sparse.

**Handling Missing Values.** In this data set, most of the values are missing (not rated). We handle this by defining the feature variable as 1,2,...,5 for the ratings and 0 for missing value. We maximize the mutual information based on the empirical distribution of values that are present, and weight it by the probability of presence for this feature. Hence, $I_{ob} = \sum_{j=1}^{n} P(X_j \neq 0) I(T; X_j | X_j \neq 0)$ and $I_{un} = E_j \{P(X_j \neq 0) I(T; X_j | X_j \neq 0)\}$. The weighting prevents 'overfitting' to movies with few ratings. Since the observed features were selected at random, the statistics of missing values of the observed and unobserved features are the same. Hence, all theorems are applicable to these definitions of $I_{ob}$ and $I_{un}$ as well.

**Greedy *IobMax* Algorithm**

We cluster the users using a simple greedy clustering algorithm . The input to the algorithm is all users, represented solely by the observed features. Since this algorithm can only find a local maximum of $I_{ob}$, we ran the algorithm 10 times (each used a different random initialization) and selected the results that had a maximum value of $I_{ob}$. More details about this algorithm can be found in [8].

In order to estimate $I_{un,D}^*$ (see definition 1), we also ran the same algorithm, where all the features are available to the algorithm (i.e. also features from group "B"). The algorithm finds clusters that maximize the mean mutual information on features from group "B".

**Results**

The results are shown in Figure 1. As $n$ increases, $I_{ob}$ decreases and $I_{un}$ increases, until they converge to each other. For small $n$, the clustering 'overfits' to the observed features. This is similar to training and test errors in supervised learning. For large $n$, $I_{un}$ approaches to $I_{un,D}^*$, which means the $IobMax$ algorithm found nearly the best possible clustering - as expected from the theorem 2. As the number of clusters increases, both $I_{ob}$ and $I_{un}$ increase, but the difference between them also increases.

## 4 Discussion and Summary

We introduce a new learning paradigm: clustering based on observed features that generalizes to unobserved features. Our results are summarized by two theorems that tell us how, without knowing the value of the unobserved features, one can estimate and maximize information between the clusters and the unobserved features.

The key assumption that enables us to prove the theorems is the *random independent* selection of the observed features. Another interpretation of the generalization theorem, without using this assumption, might be combinatorial. The difference between the observed and unobserved information is large only for a small portion of all possible partitions into observed and unobserved features. This means that almost any arbitrary partition generalizes well.

The importance of clustering which preserves information on unobserved features is that it enables us to learn new - previously unobserved - attributes from a small number of examples. Suppose that after clustering fruits based on their observed features, we eat a chinaberry[1] and thus, we "observe" (by getting sick), the previously unobserved attribute of toxicity. Assuming that in each cluster, all fruits have similar unobserved attributes, we can conclude that all fruits in the same cluster, i.e. all chinaberries, are likely to be poisonous.

We can even relate the *IobMax* principle to cognitive clustering in sensory information processing. In general, a symbolic representation (e.g. assigning object names in language) may be based on a similar principle - find a representation (clusters) that contain significant information on as many observed features as possible, while still remaining simple. Such representations are expected to contain information on other rarely viewed salient features.

### Acknowledgments

We thank Amir Globerson, Ran Bachrach, Amir Navot, Oren Shriki, Avner Dor and Ilan Sutskover for helpful discussions. We also thank the GroupLens Research Group at the University of Minnesota for use of the MovieLens data set. Our work is partly supported by grant from the Israeli Academy of Science.

## Footnotes

[1]Chinaberries are the fruits of the Melia azedarach tree, and are poisonous.

### References

[1] A. K. Jain, M. N. Murty, and P. J. Flynn. Data clustering: a review. *ACM Computing Surveys*, 31(3):264–323, September 1999.

[2] T. M. Cover and J. A. Thomas. *Elements Of Information Theory*. Wiley Interscience, 1991.

[3] V. N. Vapnik. *Statistical Learning Theory*. Wiley, 1998.

[4] N. Tishby, F. Pereira, and W. Bialek. The information bottleneck method. *Proc. 37th Allerton Conf. on Communication and Computation*, 1999.

[5] M. Seeger. Learning with labeled and unlabeled data. Technical report, University of Edinburgh, 2002.

[6] M. Szummer and T. Jaakkola. Information regularization with partially labeled data. In *NIPS*, 2003.

[7] W. Hoeffding. Probability inequalities for sums of bounded random variables. *Journal of the American Statistical Association*, 58:13–30, 1963.

[8] E. Krupka and N. Tishby. Generalization in clustering with unobserved features. Technical report, Hebrew University, 2005. http://www.cs.huji.ac.il/~tishby/nips2005tr.pdf.

[9] L. Paninski. Estimation of entropy and mutual information. *Neural Computation*, 15:1101–1253, 2003.

[10] B. Marlin. Collaborative filtering: A machine learning perspective. Master's thesis, University of Toronto, 2004.

